# Computing Robust Counter-Strategies

**Michael Johanson**
johanson@cs.ualberta.ca

**Martin Zinkevich**
maz@cs.ualberta.ca

**Michael Bowling**
Computing Science Department
University of Alberta
Edmonton, AB Canada T6G2E8
bowling@cs.ualberta.ca

## Abstract

Adaptation to other initially unknown agents often requires computing an effective counter-strategy. In the Bayesian paradigm, one must find a good counter-strategy to the inferred posterior of the other agents' behavior. In the experts paradigm, one may want to choose experts that are good counter-strategies to the other agents' expected behavior. In this paper we introduce a technique for computing *robust* counter-strategies for adaptation in multiagent scenarios under a variety of paradigms. The strategies can take advantage of a suspected tendency in the decisions of the other agents, while bounding the worst-case performance when the tendency is not observed. The technique involves solving a modified game, and therefore can make use of recently developed algorithms for solving very large extensive games. We demonstrate the effectiveness of the technique in two-player Texas Hold'em. We show that the computed poker strategies are substantially more robust than best response counter-strategies, while still exploiting a suspected tendency. We also compose the generated strategies in an experts algorithm showing a dramatic improvement in performance over using simple best responses.

## 1   Introduction

Many applications for autonomous decision making (e.g., assistive technologies, electronic commerce, interactive entertainment) involve other agents interacting in the same environment. The agents' choices are often not independent, and good performance may necessitate adapting to the behavior of the other agents. A number of paradigms have been proposed for adaptive decision making in multiagent scenarios. The agent modeling paradigm proposes to learn a predictive model of other agents' behavior from observations of their decisions. The model is then used to compute or select a counter-strategy that will perform well given the model. An alternative paradigm is the mixture of experts. In this approach, a set of expert strategies is identified a priori. These experts can be thought of as counter-strategies for the range of expected tendencies in the other agents' behavior. The decision maker, then, chooses amongst the counter-strategies based on their online performance, commonly using techniques for regret minimization (e.g., UCB1 [ACBF02]). In either approach, finding counter-strategies is an important subcomponent.

The most common approach to choosing a counter-strategy is best response: the performance maximizing strategy if the other agents' behavior is known [Rob51, CM96]. In large domains where best response computations are not tractable, they are often approximated with "good responses" from a computationally tractable set, where performance maximization remains the only criterion [RV02]. The problem with this approach is that best response strategies can be very brittle. While max-

imizing performance against the model, they can (and often do) perform poorly when the model is wrong. The use of best response counter-strategies, therefore, puts an impossible burden on a priori choices, either the agent model bias or the set of expert counter-strategies. McCracken and Bowling [MB04] proposed $\epsilon$-safe strategies to address this issue. Their technique chooses the best performance maximizing strategy from the set of strategies that don't lose more than $\epsilon$ in the worst-case. The strategy balances exploiting the agent model with a safety guarantee in case the model is wrong. Although conceptually appealing, it is computationally infeasible even for moderately sized domains and has only been employed in the simple game of Ro-Sham-Bo.

In this paper, we introduce a new technique for computing robust counter-strategies. The counter-strategies, called *restricted Nash responses*, balance performance maximization against the model with reasonable performance even when the model is wrong. The technique involves computing a Nash equilibrium of a modified game, and therefore can exploit recent advances in solving large extensive games [GHPS07, ZBB07, ZJBP08]. We demonstrate the practicality of the approach in the challenging domain of poker. We begin by reviewing the concepts of extensive form games, best responses, and Nash equilibria, as well as describing how these concepts apply in the poker domain. We then describe a technique for computing an approximate best response to an arbitrary poker strategy, and show that this, indeed, produces brittle counter-strategies. We then introduce restricted Nash responses, describe how they can be computed efficiently, and show that they are significantly more robust while still being effective counter-strategies. Finally, we demonstrate that these strategies can be used in an experts algorithm to make a more effective adaptive player than when using simple best response.

## 2   Background

A **perfect information extensive game** consists of a tree of game states. At each game state, an action is made either by nature, or by one of the players, or the state is a terminal state where each player receives a fixed utility. A strategy for a player consists of a distribution over actions for every game state. In an **imperfect information extensive game**, the states where a player makes an action are divided into information sets. When a player chooses an action, it does not know the state of the game, only the information set, and therefore its strategy is a mapping from information sets to distributions over actions. A common restriction on imperfect information extensive games is **perfect recall**, where two states can only be in the same information set for a player if that player took the same actions from the same information sets to reach the two game states. In the remainder of the paper, we will be considering imperfect information extensive games with perfect recall.

Let $\sigma_i$ be a strategy for player $i$ where $\sigma_i(I, a)$ is the probability that strategy assigns to action $a$ in information set $I$. Let $\Sigma_i$ be the set of strategies for player $i$, and define $u_i(\sigma_1, \sigma_2)$ to be the expected utility of player $i$ if player 1 uses $\sigma_1 \in \Sigma_1$ and player 2 uses $\sigma_2 \in \Sigma_2$. Define $BR(\sigma_2) \subseteq \Sigma_1$ to be the set of best responses to $\sigma_2$, i.e.:

$$BR(\sigma_2) = \operatorname*{argmax}_{\sigma_1 \in \Sigma_1} u_1(\sigma_1, \sigma_2) \tag{1}$$

and define $BR(\sigma_1) \subseteq \Sigma_2$ similarly. If $\sigma_1 \in BR(\sigma_2)$ and $\sigma_2 \in BR(\sigma_1)$, then $(\sigma_1, \sigma_2)$ is a **Nash equilibrium**. A **zero-sum extensive game** is an extensive game where $u_1 = -u_2$. In this type of game, for any two equilibria $(\sigma_1, \sigma_2)$ and $(\sigma'_1, \sigma'_2)$, $u_1(\sigma_1, \sigma_2) = u_1(\sigma'_1, \sigma'_2)$ and $(\sigma_1, \sigma'_2)$ (as well as $(\sigma'_1, \sigma_2)$) are also equilibria. Define the **value of the game to player 1** ($v_1$) to be the expected utility of player 1 in equilibrium. In a zero-sum extensive game, the **exploitability** of a strategy $\sigma_1 \in \Sigma_1$ is:

$$\operatorname{ex}(\sigma_1) = \max_{\sigma_2 \in \Sigma_2} (v_1 - u_1(\sigma_1, \sigma_2)). \tag{2}$$

The value of the game to player 2 ($v_2$) and the exploitability of a strategy $\sigma_2 \in \Sigma_2$ are defined similarly. A strategy which can be exploited for no more than $\epsilon$ is $\epsilon$-**safe**. An $\epsilon$-**Nash equilibrium** in a zero-sum extensive game is a strategy pair where both strategies are $\epsilon$-safe.

In the remainder of the work, we will be dealing with mixing two strategies. Informally, one can think of mixing two strategies as performing the following operation: first, flip a (possibly biased) coin; if it comes up heads, use the first strategy, otherwise use the second strategy. Formally, define $\pi^{\sigma_i}(I)$ to be the probability that player $i$ when following strategy $\sigma_i$ chooses the actions necessary to

make information set $I$ reachable from the root of the game tree. Given $\sigma_1, \sigma_1' \in \Sigma_1$ and $p \in [0,1]$, define $\mathrm{mix}_p(\sigma_1, \sigma_1') \in \Sigma_1$ such that for any information set $I$ of player 1, for all actions $a$:

$$\mathrm{mix}_p(\sigma_1, \sigma_1')(I, a) = \frac{p \times \pi^{\sigma_1}(I)\sigma_1(I, a) + (1-p) \times \pi^{\sigma_1'}(I)\sigma_1(I, a)}{p \times \pi^{\sigma_1}(I) + (1-p) \times \pi^{\sigma_1'}(I)}. \quad (3)$$

Given an event $E$, define $\Pr_{\sigma_1, \sigma_2}[E]$ to be the probability of the event $E$ given player 1 uses $\sigma_1$, and player 2 uses $\sigma_2$. Given the above definition of mix, it is the case that for all $\sigma_1, \sigma_1' \in \Sigma_1$, all $\sigma_2 \in \Sigma_2$, all $p \in [0,1]$, and all events $E$:

$$\Pr_{\mathrm{mix}_p(\sigma_1, \sigma_1'), \sigma_2}[E] = p \Pr_{\sigma_1, \sigma_2}[E] + (1-p) \Pr_{\sigma_1', \sigma_2}[E] \quad (4)$$

So probabilities of outcomes can simply be combined linearly. As a result the utility of a mixture of strategies is just $u(\mathrm{mix}_p(\sigma_1, \sigma_1'), \sigma_2) = pu(\sigma_1, \sigma_2) + (1-p)u(\sigma_1', \sigma_2)$.

## 3  Texas Hold'Em

While the techniques in this paper apply to general extensive games, our empirical results will focus on the domain of poker. In particular, we look at heads-up limit Texas Hold'em, the game used in the AAAI Computer Poker Competition [ZL06]. A single hand of this poker variant consists of two players each being dealt two private cards, followed by five community cards being revealed. Each player tries to form the best five-card poker hand from the community cards and her private cards: if the hand goes to a showdown, the player with the best five-card hand wins the pot. The key to good play is on average to have more chips in the pot when you win than are in the pot when you lose. The players' actions control the pot size through betting. After the private cards are dealt, a round of betting occurs, followed by additional betting rounds after the third (flop), fourth (turn), and fifth (river) community cards are revealed. Betting rounds involve players alternately deciding to either fold (letting the other player win the chips in the pot), call (matching the opponent's chips in the pot), or raise (matching, and then adding an additional fixed amount into the pot). No more than four raises are allowed in a single betting round. Notice that heads-up limit Texas Hold'em is an example of a finite imperfect information extensive game with perfect recall. When evaluating the results of a match (several hands of poker) between two players, we find it convenient to state the result in millibets won per hand. A millibet is one thousandth of a small-bet, the fixed magnitude of bets used in the first two rounds of betting. To provide some intuition for these numbers, a player that always folds will lose 750 mb/h while a typical player that is 10 mb/h stronger than another would require over one million hands to be 95% certain to have won overall.

**Abstraction.**  While being a relatively small variant of poker, the game tree for heads-up limit Texas Hold'em is still very large, having approximately $9.17 \times 10^{17}$ states. Fundamental operations, such as computing a best response strategy or a Nash equilibrium as described in Section 2, are intractable on the full game. Common practice is to define a more reasonably sized abstraction by merging information sets (e.g., by treating certain hands as indistinguishable). If the abstraction involves the same betting structure, a strategy for an abstract game can be played directly in the full game. If the abstraction is small enough Nash equilibria and best response computations become feasible. Finding an approximate Nash equilibrium in an abstract game has proven to be an effective way to construct a strong program for the full game [BBD+03, GS06]. Recent solution techniques have been able to compute approximate Nash equilibria for abstractions with as many as $10^{10}$ game states [ZBB07, GHPS07]. Given a strategy defined in a small enough abstraction, it is also possible to compute a best response to the strategy in the abstract game. This can be done in time linear in the size of the extensive game. The abstraction used in this paper has approximately $6.45 \times 10^9$ game states, and is described in an accompanying technical report [JZB07].

**The Competitors.**  Since this work focuses on adapting to other agents' behavior, our experiments make use of a battery of different poker playing programs. We give a brief description of these programs here. **PsOpti4** [BBD+03] is one of the earliest successful near equilibrium programs for poker and is available as "Sparbot" in the commercial title *Poker Academy*. **PsOpti6** is a later and weaker variant, but whose weaknesses are thought to be less obvious to human players. Together, PsOpti4 and PsOpti6 formed **Hyperborean**, the winner of the AAAI 2006 Computer Poker Competition. **S1239**, **S1399**, and **S2298** are similar near equilibrium strategies generated by a new

equilibrium computation method [ZBB07] using a much larger abstraction than is used in PsOpti4 and PsOpti6. **A60** and **A80** are two past failed attempts at generating interesting exploitive strategies, and are highly exploitable for over 1000 mb/h. **CFR5** is a new near Nash equilibrium [ZJBP08], and uses the abstraction described in the accompanying technical report [JZB07]. We will also experiment with two programs **Bluffbot** and **Monash**, who placed second and third respectively in the AAAI 2006 Computer Poker Competition's bankroll event [ZL06].

## 4  Frequentist Best Response

In the introduction, we described best response counter-strategies as brittle, performing poorly when playing against a different strategy from the one which they were computed to exploit. In this section, we examine this claim empirically in the domain of poker. Since a best response computation is intractable in the full game, we first describe a technique, called frequentist best response, for finding a "good response" using an abstract game. As described in the previous section, given a strategy in an abstract game we can compute a best response to that strategy within the abstraction. The challenge is that the abstraction used by an arbitrary opponent is not known. In addition, it may be beneficial to find a best response in an alternative, possible more powerful, abstraction.

Suppose we want to find a "good response" to some strategy P. The basic idea of frequentist best response (FBR) is to observe P playing the full game of poker, construct a model of it in an abstract game (unrelated to that P's own abstraction), and then compute a best-response in this abstraction. FBR first needs many examples of the strategy playing the full, unabstracted game. It then iterates through every one of P's actions for every hand. It finds the action's associated information set in the abstract game and increments a counter associated with that information set and action. After observing a sufficient number of hands, we can construct a strategy in the abstract game based on the frequency counts. At each information set, we set the strategy's probability for performing each action to be the number of observations of that action being chosen from that information set, divided by the total number of observations in the information set. If an information set was never observed, the strategy defaults to the call action. Since this strategy is defined in a known abstraction, FBR can simply calculate a best response to this frequentist strategy.

P's opponent in the observed games greatly affects the quality of the model. We have found it most effective to have P play against a trivial strategy that calls and raises with equal probability. This provides with us the most observations of P's decisions that are well distributed throughout the possible betting sequences. Observing P in self-play or against near equilibrium strategies has shown to require considerably more observed hands. We typically use 5 million hands of training data to compute the model strategy, although reasonable responses can still be computed with as few as 1 million hands.

**Evaluation.**   We computed frequentist best response strategies against seven different opponents. We played the resulting responses both against the opponent it was designed to exploit as well as the other six opponents and an approximate equilibrium strategy computed using the same abstraction. The results of this tournament are shown as a crosstable in Table 1. Positive numbers (appearing with a green background) are in favor of the row player (FBR strategies, in this case).

The first thing to notice is that FBR is very successful at exploiting the opponent it was designed to exploit, i.e., the diagonal of the crosstable is positive and often large. In some cases, FBR identified strategies exploiting the opponent for more than previously known to be possible, e.g., PsOpti4 had only previously been exploited for 75 mb/h [Sch06], while FBR exploits it for 137 mb/h. The second thing to notice is that when FBR strategies play against other opponents their performance is poor, i.e., the off-diagonal of the crosstable is generally negative and occasionally by a large amount. For example, A60 is not a strong program. It is exploitable for over 2000 mb/h (note that always fold only loses 750 mb/h) and an approximate equilibrium strategy defeats it by 93 mb/h. Yet, every FBR strategy besides the one trained on it, loses to it, sometimes by a substantial amount. These results give evidence that best response is, in practice, a brittle computation, and can perform poorly when the model is wrong.

One exception to this trend is play within the family of S-bots. In particular, consider S1399 and S1239, which are very similar programs, using the same technique for equilibrium computation with the same abstract game. They only differ in the number of iterations the algorithm was afforded. The

| | Opponents | | | | | | | | Average |
|---|---|---|---|---|---|---|---|---|---|
| | PsOpti4 | PsOpti6 | A60 | A80 | S1239 | S1399 | S2298 | CFR5 | |
| FBR-PsOpti4 | 137 | -163 | -227 | -231 | -106 | -85 | -144 | -210 | -129 |
| FBR-PsOpti6 | -79 | 330 | -68 | -89 | -36 | -23 | -48 | -97 | -14 |
| FBR-A60 | -442 | -499 | 2170 | -701 | -359 | -305 | -377 | -620 | -142 |
| FBR-A80 | -312 | -281 | -557 | 1048 | -251 | -231 | -266 | -331 | -148 |
| FBR-S1239 | -20 | 105 | -89 | -42 | 106 | 91 | -32 | -87 | 3 |
| FBR-S1399 | -43 | 38 | -48 | -77 | 75 | 118 | -46 | -109 | -11 |
| FBR-S2298 | -39 | 51 | -50 | -26 | 42 | 50 | 33 | -41 | 2 |
| CFR5 | 36 | 123 | 93 | 41 | 70 | 68 | 17 | 0 | 56 |
| Max | 137 | 330 | 2170 | 1048 | 106 | 118 | 33 | 0 | |

Table 1: Results of frequentist best responses (FBR) against a variety of opponent programs in full Texas Hold'em, with winnings in mb/h for the row player. Results involving PsOpti4 or PsOpti6 used 10 duplicate matches of 10,000 hands and are significant to 20 mb/h. Other results used 10 duplicate matches of 500,000 hands and are significant to 2 mb/h.

results show they do share weaknesses as FBR-S1399 does beat S1239 by 75 mb/h. However, this is 30% less than 106 mb/h, the amount that FBR-S1239 beats the same opponent. Considering the similarity of these opponents, even this apparent exception is actually suggestive that best response is not robust to even slight changes in the model.

Finally, consider the performance of the approximate equilibrium player, CFR5. As it was computed from a relatively large abstraction it performs comparably well, not losing to any of the seven opponents. However, it also does not win by the margins of the correct FBR strategy. As noted, against the highly exploitable A60, it wins by a mere 93 mb/h. What we really want is a compromise. We would like a strategy that can exploit an opponent successfully like FBR, but without the large penalty when playing against a different opponent. The remainder of the paper examines Restricted Nash Response, a technique for creating such strategies.

## 5 Restricted Nash Response

Imagine that you had a model of your opponent, but did not believe that this model was perfect. The model may capture the general idea of the adversary you expect to face, but most likely is not identical. For example, maybe you have played a previous version of the same program, have a model of its play, but suspect that the designer is likely to have made some small improvements in the new version. One way to explicitly define our situation is that with the new version we might expect that 75 percent of the hands will be played identically to the old version. The other 25 percent is some new modification, for which we want to be robust. This, in itself, can be thought of as a game for which we can apply the usual game theoretic machinery of equilibria.

Let our model of our opponent be some strategy $\sigma_{\text{fix}} \in \Sigma_2$. Define $\Sigma_2^{p,\sigma_{\text{fix}}}$ to be those strategies of the form $\text{mix}_p(\sigma_{\text{fix}}, \sigma_2')$, where $\sigma_2'$ is an arbitrary strategy in $\Sigma_2$. Define the set of **restricted best responses to $\sigma_1 \in \Sigma_1$** to be:

$$BR^{p,\sigma_{\text{fix}}}(\sigma_1) = \underset{\sigma_2 \in \Sigma_2^{p,\sigma_{\text{fix}}}}{\text{argmax}} \, u_2(\sigma_1, \sigma_2) \tag{5}$$

A **$(p, \sigma_{\text{fix}})$ restricted Nash equilibrium** is a pair of strategies $(\sigma_1^*, \sigma_2^*)$ where $\sigma_2^* \in BR^{p,\sigma_{\text{fix}}}(\sigma_1^*)$ and $\sigma_1^* \in BR(\sigma_2^*)$. In this pair, the strategy $\sigma_1^*$ is a **$p$-restricted Nash response (RNR) to $\sigma_{\text{fix}}$**. We propose these RNRs would be ideal counter-strategies for $\sigma_{\text{fix}}$, where $p$ provides a balance between exploitation and exploitability. This concept is closely related to $\epsilon$-safe best responses [MB04]. Define $\Sigma_1^{\epsilon\text{-safe}} \subseteq \Sigma_1$ to be the set of all strategies which are $\epsilon$-safe (with an exploitability less than $\epsilon$). Then the set of $\epsilon$-safe best responses are:

$$BR^{\epsilon\text{-safe}}(\sigma_2) = \underset{\sigma_1 \in \Sigma^{\epsilon\text{-safe}}}{\text{argmax}} \, u_1(\sigma_1, \sigma_2) \tag{6}$$

**Theorem 1** *For all $\sigma_2 \in \Sigma_2$, for all $p \in (0, 1]$, if $\sigma_1$ is a $p$-RNR to $\sigma_2$, then there exists an $\epsilon$ such that $\sigma_1$ is an $\epsilon$-safe best response to $\sigma_2$.*

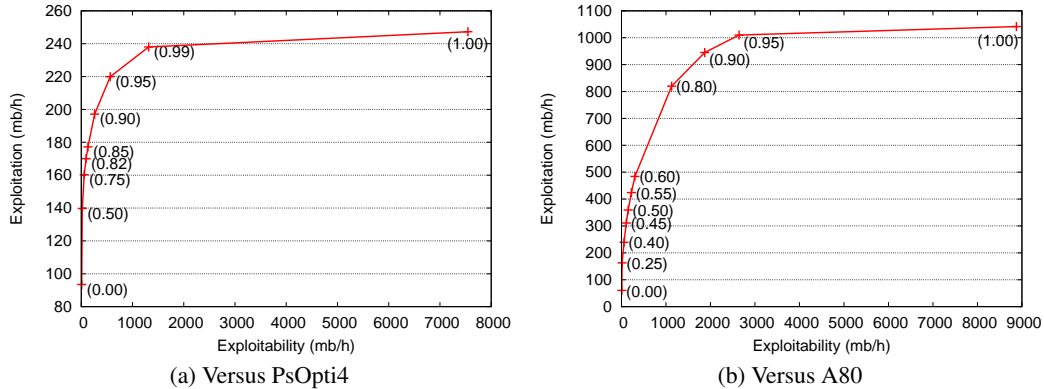

(a) Versus PsOpti4       (b) Versus A80

Figure 1: The tradeoff between $\epsilon$ and utility. For each opponent, we varied $p \in [0, 1]$ for the RNR. The labels at each datapoint indicate the value of $p$ used.

The proof of Theorem 1 is in an accompanying technical report [JZB07]. The significance of Theorem 1 is that, among all strategies that are at most $\epsilon$ suboptimal, the RNR strategies are among the best responses. Thus, if we want a strategy that is at most $\epsilon$ suboptimal, we can vary $p$ to produce a strategy that is the best response among all such $\epsilon$-safe strategies.

Unlike safe best responses, a RNR can be computed by just solving a modification of the original abstract game. For example, if using a sequence form representation of linear programming then one just needs to add lower bound constraints for the restricted player's realization plan probabilities. In our experiments we use a recently developed solution technique based on regret minimization [ZJBP08] with a modified game that starts with an unobserved chance node deciding whether the restricted player is forced to use strategy $\sigma_{\text{fix}}$ on the current hand. The RNRs used in our experiments were computed with less than a day of computation on a 2.4Ghz AMD Opteron.

**Choosing p.**  In order to compute a RNR we have to choose a value of $p$. By varying the value $p \in [0, 1]$, we can produce poker strategies that are closer to a Nash equilibrium (when p is near 0) or are closer to the best response (when p is near 1). When producing an RNR to a particular opponent, it is useful to consider the tradeoff between the utility of the response against that opponent and the exploitability of the response itself. We explore this tradeoff in Figure 1. In 1a we plot the results of using RNR with various values of $p$ against the model of PsOpti4. The x-axis shows the exploitability of the response, $\epsilon$. The y-axis shows the exploitation of the model by the response in the abstract game. Note that the actual exploitation and exploitability in the full game may be different, as we explore later. Figure 1b shows this tradeoff against A80.

Notice that by selecting values of p, we can control the tradeoff between $\epsilon$ and the response's exploitation of the strategy. More importantly, the curves are highly concave meaning that dramatic reductions in exploitability can be achieved with only a small sacrifice in the ability to exploit the model.

**Evaluation.**  We used RNR to compute a counter-strategy to the same seven opponents used in the FBR experiments, with the $p$ value used for each opponent selected such that the resulting $\epsilon$ is close to 100 mb/h. The RNR strategies were played against these seven opponents and the equilibrium CFR5 in the full game of Texas Hold'em. The results of this tournament are displayed as a crosstable in Table 2.

The first thing to notice is that RNR is capable of exploiting the opponent for which it was designed as a counter-strategy, while still performing well against the other opponents. In other words, not only is the diagonal positive and large, most of the crosstable is positive. For the highly exploitable opponents, such as A60 and A80, the degree of exploitation is much reduced from FBR, which is a consequence of choosing $p$ such that $\epsilon$ is at most 100 mb/h. Notice, though, that it does exploit these opponents significantly more than the approximate Nash strategy (CFR5).

|  | Opponents | | | | | | | | Average |
|---|---|---|---|---|---|---|---|---|---|
|  | PsOpti4 | PsOpti6 | A60 | A80 | S1239 | S1399 | S2298 | CFR5 |  |
| RNR-PsOpti4 | 85 | 112 | 39 | 9 | 63 | 61 | -1 | -23 | 43 |
| RNR-PsOpti6 | 26 | 234 | 72 | 34 | 59 | 59 | 1 | -28 | 57 |
| RNR-A60 | -17 | 63 | 582 | -22 | 37 | 39 | -9 | -45 | 78 |
| RNR-A80 | -7 | 66 | 22 | 293 | 11 | 12 | 0 | -29 | 46 |
| RNR-S1239 | 38 | 130 | 68 | 31 | 111 | 106 | 9 | -20 | 59 |
| RNR-S1399 | 31 | 136 | 66 | 29 | 105 | 112 | 6 | -24 | 58 |
| RNR-S2298 | 21 | 137 | 72 | 30 | 77 | 76 | 31 | -11 | 54 |
| CFR5 | 36 | 123 | 93 | 41 | 70 | 68 | 17 | 0 | 56 |
| Max | 85 | 234 | 582 | 293 | 111 | 112 | 31 | 0 |  |

Table 2: Results of restricted Nash response (RNR) against a variety of opponent programs in full Texas Hold'em, with winnings in mb/h for the row player. See the caption of Table 1 for match details.

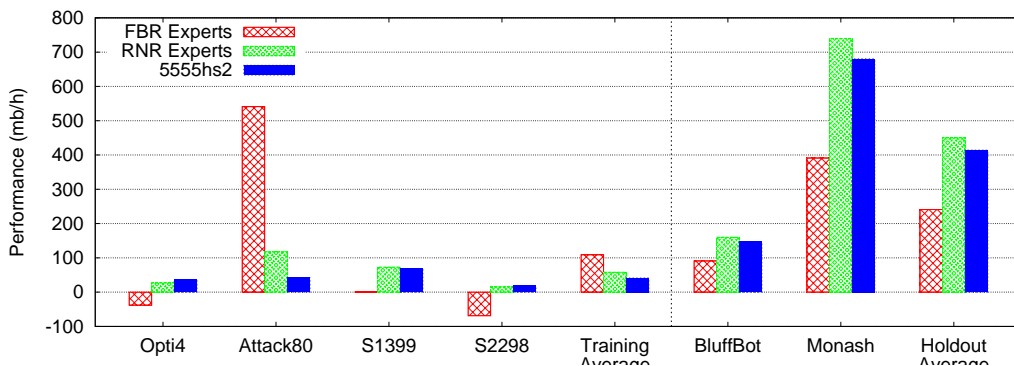

Figure 2: Performance of FBR-experts, RNR-experts, and a near Nash equilibrium strategy (CFR5) against "training" opponents and "hold out" opponents in 50 duplicate matches of 1000 hands.

Revisiting the family of S-bots, we notice that the known similarity of S1239 and S1399 is more apparent with RNR. The performance of RNR with the correct model against these two players is close to that of FBR, while the performance with the similar model is only a 6mb/h drop. Essentially, RNR is forced to exploit only the weaknesses that are general and is robust to small changes. Overall, RNR offers a similar degree of exploitation to FBR, but with far more robustness.

## 6   Restricted Nash Experts

We have shown that RNR can be used to find robust counter-strategies. In this section we investigate their use in an adaptive poker program. We generated four counter-strategies based on the opponents PsOpti4, A80, S1399, and S2298, and then used these as experts which UCB1 [ACBF02] (a regret minimizing algorithm) selected amongst. The FBR-experts algorithm used a FBR to each opponent, and the RNR-experts used RNR to each opponent. We then played these two expert mixtures in 1000 hand matches against both the four programs used to generate the counter strategies as well as two programs from the 2006 AAAI Computer Poker Competition, which have an unknown origin and were developed independently of the other programs. We call the first four programs "training opponents" and the other two programs "holdout opponents", as they are similar to training error and holdout error in supervised learning.

The results of these matches are shown in Figure 2. As expected, when the opponent matches one of the training models, FBR-experts and RNR-experts perform better, on average, than a near equilibrium strategy (see "Training Average" in Figure 2). However, if we look at the break down against individual opponents, we see that all of FBR's performance comes from its ability to significantly exploit one single opponent. Against the other opponents, it actually performs worse than the non-adaptive near equilibrium strategy. RNR does not exploit A80 to the same degree as FBR, but also does not lose to any opponent.

The comparison with the holdout opponents, though, is more realistic and more telling. Since it is unlikely a player will have a model of the exact program its likely to face in a competition, it is important for its counter-strategies to exploit general weaknesses that might be encountered. Our holdout programs have no explicit relationship to the training programs, yet the RNR counter-strategies are still effective at exploiting these programs as demonstrated by the expert mixture being able to exploit these programs by more than the near equilibrium strategy. The FBR counter-strategies, on the other hand, performed poorly outside of the training programs, demonstrating once again that RNR counter-strategies are both more robust and more suitable as a basis for adapting behavior to other agents in the environment.

## 7  Conclusion

We proposed a new technique for generating *robust* counter-strategies in multiagent scenarios. The restricted Nash responses balance exploiting suspected tendencies in other agents' behavior, while bounding the worst-case performance when the tendency is not observed. The technique involves computing an approximate equilibrium to a modification of the original game, and therefore can make use of recently developed algorithms for solving very large extensive games. We demonstrated the technique in the domain of poker, showing it to generate more robust counter-strategies than traditional best response. We also showed that a simple mixture of experts algorithm based on restricted Nash response counter-strategies was far superior to using best response counter-strategies if the exact opponent was not used in training. Further, the restricted Nash experts algorithm outperformed a static non-adaptive near equilibrium at exploiting the previously unseen programs.

## References

[ACBF02]  P. Auer, N. Cesa-Bianchi, and P. Fischer. Finite time analysis of the multiarmed bandit problem. *Machine Learning*, 47:235–256, 2002.

[BBD+03]  D. Billings, N. Burch, A. Davidson, R. Holte, J. Schaeffer, T. Schauenberg, and D. Szafron. Approximating game-theoretic optimal strategies for full-scale poker. In *International Joint Conference on Artificial Intelligence*, pages 661–668, 2003.

[CM96]  David Carmel and Shaul Markovitch. Learning models of intelligent agents. In *Proceedings of the Thirteenth National Conference on Artificial Intelligence*, Menlo Park, CA, 1996. AAAI Press.

[GHPS07]  A. Gilpin, S. Hoda, J. Pena, and T. Sandholm. Gradient-based algorithms for finding nash equilibria in extensive form games. In *Proceedings of the Eighteenth International Conference on Game Theory*, 2007.

[GS06]  A. Gilpin and T. Sandholm. A competitive texas hold'em poker player via automated abstraction and real-time equilibrium computation. In *National Conference on Artificial Intelligence*, 2006.

[JZB07]  Michael Johanson, Martin Zinkevich, and Michael Bowling. Computing robust counter-strategies. Technical Report TR07-15, Department of Computing Science, University of Alberta, 2007.

[MB04]  Peter McCracken and Michael Bowling. Safe strategies for agent modelling in games. In *AAAI Fall Symposium on Artificial Multi-agent Learning*, October 2004.

[Rob51]  Julia Robinson. An iterative method of solving a game. *Annals of Mathematics*, 54:296–301, 1951.

[RV02]  Patrick Riley and Manuela Veloso. Planning for distributed execution through use of probabilistic opponent models. In *Proceedings of the Sixth International Conference on AI Planning and Scheduling*, pages 77–82, April 2002.

[Sch06]  T.C. Schauenberg. Opponent modelling and search in poker. Master's thesis, University of Alberta, 2006.

[ZBB07]  M. Zinkevich, M. Bowling, and N. Burch. A new algorithm for generating strong strategies in massive zero-sum games. In *Proceedings of the Twenty-Seventh Conference on Artificial Intelligence (AAAI)*, 2007. To Appear.

[ZJBP08]  M. Zinkevich, M. Johanson, M. Bowling, and C. Piccione. Regret minimization in games with incomplete information. In *Neural Information Processing Systems 21*, 2008.

[ZL06]  M. Zinkevich and M. Littman. The AAAI computer poker competition. *Journal of the International Computer Games Association*, 29, 2006. News item.

